# Near-optimal Regret Bounds for Reinforcement Learning

**Peter Auer**     **Thomas Jaksch**     **Ronald Ortner**

University of Leoben, Franz-Josef-Strasse 18, 8700 Leoben, Austria

{auer,tjaksch,rortner}@unileoben.ac.at

## Abstract

For undiscounted reinforcement learning in Markov decision processes (MDPs) we consider the *total regret* of a learning algorithm with respect to an optimal policy. In order to describe the transition structure of an MDP we propose a new parameter: An MDP has *diameter $D$* if for any pair of states $s, s'$ there is a policy which moves from $s$ to $s'$ in at most $D$ steps (on average). We present a reinforcement learning algorithm with total regret $\tilde{O}(DS\sqrt{AT})$ after $T$ steps for any unknown MDP with $S$ states, $A$ actions per state, and diameter $D$. This bound holds with high probability. We also present a corresponding lower bound of $\Omega(\sqrt{DSAT})$ on the total regret of any learning algorithm.

## 1   Introduction

In a Markov decision process (MDP) $M$ with finite state space $\mathcal{S}$ and finite action space $\mathcal{A}$, a learner in state $s \in \mathcal{S}$ needs to choose an action $a \in \mathcal{A}$. When executing action $a$ in state $s$, the learner receives a random reward $r$ with mean $\bar{r}(s, a)$ according to some distribution on $[0, 1]$. Further, according to the transition probabilities $p(s'|s, a)$, a random transition to a state $s' \in \mathcal{S}$ occurs.

Reinforcement learning of MDPs is a standard model for learning with delayed feedback. In contrast to important other work on reinforcement learning — where the performance of the *learned* policy is considered (see e.g. [1, 2] and also the discussion and references given in the introduction of [3]) — we are interested in the performance of the learning algorithm *during learning*. For that, we compare the rewards collected by the algorithm during learning with the rewards of an optimal policy.

In this paper we will consider *undiscounted* rewards. The *accumulated reward* of an algorithm $\mathfrak{A}$ after $T$ steps in an MDP $M$ is defined as

$$R(M, \mathfrak{A}, s, T) := \sum_{t=1}^{T} r_t,$$

where $s$ is the initial state and $r_t$ are the rewards received during the execution of algorithm $\mathfrak{A}$. The *average reward*

$$\rho(M, \mathfrak{A}, s) := \lim_{T \to \infty} \frac{1}{T} \, \mathbb{E}\left[R(M, \mathfrak{A}, s, T)\right]$$

can be maximized by an appropriate stationary *policy* $\pi : \mathcal{S} \to \mathcal{A}$ which defines an optimal action for each state [4].

The difficulty of learning an MDP does not only depend on its size (given by the number of states and actions), but also on its transition structure. In order to measure this transition structure we propose a new parameter, the *diameter $D$* of an MDP. The diameter $D$ is the time it takes to move from any state $s$ to any other state $s'$, using an appropriate policy for this pair of states $s$ and $s'$:

**Definition 1.** *Let $T(s'|M, \pi, s)$ be the first (random) time step in which state $s'$ is reached when policy $\pi$ is executed on MDP $M$ with initial state $s$. Then the* diameter *of $M$ is given by*

$$D(M) := \max_{s, s' \in \mathcal{S}} \min_{\pi: \mathcal{S} \to \mathcal{A}} \mathbb{E}\left[T(s'|M, \pi, s)\right].$$

A finite diameter seems necessary for interesting bounds on the *regret* of any algorithm with respect to an optimal policy. When a learner explores suboptimal actions, this may take him into a "bad part" of the MDP from which it may take about $D$ steps to reach again a "good part" of the MDP. Hence, the learner may suffer regret $D$ for such exploration, and it is very plausible that the diameter appears in the regret bound.

For MDPs with finite diameter (which usually are called *communicating*, see e.g. [4]) the optimal average reward $\rho^*$ does not depend on the initial state (cf. [4], Section 8.3.3), and we set

$$\rho^*(M) := \rho^*(M, s) := \max_\pi \rho(M, \pi, s).$$

The optimal average reward is the natural benchmark for a learning algorithm $\mathfrak{A}$, and we define the *total regret* of $\mathfrak{A}$ after $T$ steps as[1]

$$\Delta(M, \mathfrak{A}, s, T) := T\rho^*(M) - R(M, \mathfrak{A}, s, T).$$

In the following, we present our reinforcement learning algorithm UCRL2 (a variant of the UCRL algorithm of [5]) which uses upper confidence bounds to choose an optimistic policy. We show that the total regret of UCRL2 after $T$ steps is $\tilde{O}(D|\mathcal{S}|\sqrt{|\mathcal{A}|T})$. A corresponding lower bound of $\Omega(\sqrt{D|\mathcal{S}||\mathcal{A}|T})$ on the total regret of any learning algorithm is given as well. These results establish the diameter as an important parameter of an MDP. Further, the diameter seems to be more natural than other parameters that have been proposed for various PAC and regret bounds, such as the *mixing time* [3, 6] or the *hitting time* of an optimal policy [7] (cf. the discussion below).

## 1.1 Relation to previous Work

We first compare our results to the PAC bounds for the well-known algorithms $E^3$ of Kearns, Singh [3], and R-Max of Brafman, Tennenholtz [6] (see also Kakade [8]). These algorithms achieve $\varepsilon$-optimal average reward with probability $1 - \delta$ after time polynomial in $\frac{1}{\delta}$, $\frac{1}{\varepsilon}$, $|\mathcal{S}|$, $|\mathcal{A}|$, and the mixing time $T_\varepsilon^{\mathrm{mix}}$ (see below). As the polynomial dependence on $\varepsilon$ is of order $1/\varepsilon^3$, the PAC bounds translate into $T^{2/3}$ regret bounds at the best. Moreover, both algorithms need the $\varepsilon$-*return mixing time* $T_\varepsilon^{\mathrm{mix}}$ of an optimal policy $\pi^*$ as input parameter. This parameter $T_\varepsilon^{\mathrm{mix}}$ is the number of steps until the average reward of $\pi^*$ over these $T_\varepsilon^{\mathrm{mix}}$ steps is $\varepsilon$-close to the optimal average reward $\rho^*$. It is easy to construct MDPs of diameter $D$ with $T_\varepsilon^{\mathrm{mix}} \approx D/\varepsilon$. This additional dependency on $\varepsilon$ further increases the exponent in the above mentioned regret bounds for $E^3$ and R-max. Also, the exponents of the parameters $|\mathcal{S}|$ and $|\mathcal{A}|$ in the PAC bounds of [3] and [6] are substantially larger than in our bound.

The MBIE algorithm of Strehl and Littman [9, 10] — similarly to our approach — applies confidence bounds to compute an optimistic policy. However, Strehl and Littman consider only a discounted reward setting, which seems to be less natural when dealing with regret. Their definition of regret measures the difference between the rewards[2] of an optimal policy and the rewards of the learning algorithm *along the trajectory taken by the learning algorithm*. In contrast, we are interested in the regret of the learning algorithm in respect to the rewards of the optimal policy *along the trajectory of the optimal policy*.

Tewari and Bartlett [7] propose a generalization of the *index policies* of Burnetas and Katehakis [11]. These index policies choose actions optimistically by using confidence bounds only for the estimates in the current state. The regret bounds for the *index policies* of [11] and the OLP algorithm of [7] are *asymptotically* logarithmic in $T$. However, unlike our bounds, these bounds depend on the gap between the "quality" of the best and the second best action, and these asymptotic bounds also hide an additive term which is exponential in the number of states. Actually, it is possible to prove a corresponding gap-dependent logarithmic bound for our UCRL2 algorithm as well (cf. Remark 4 below). This bound holds uniformly over time and under weaker assumptions: While [7] and [11] consider only *ergodic* MDPs in which *any* policy will reach every state after a sufficient number of steps, we make only the more natural assumption of a finite diameter.

## 2 Results

We summarize the results achieved for our algorithm UCRL2 which is described in the next section, and also state a corresponding lower bound. We assume an unknown MDP $M$ to be learned, with $S := |\mathcal{S}|$ states, $A := |\mathcal{A}|$ actions, and finite diameter $D := D(M)$. Only $\mathcal{S}$ and $\mathcal{A}$ are known to the learner, and UCRL2 is run with parameter $\delta$.

**Theorem 2.** *With probability $1 - \delta$ it holds that for any initial state $s \in \mathcal{S}$ and any $T > 1$, the regret of UCRL2 is bounded by*

$$\Delta(M, \text{UCRL2}, s, T) \ \leq \ c_1 \cdot DS \sqrt{TA \log \tfrac{T}{\delta}},$$

*for a constant $c_1$ which is independent of $M$, $T$, and $\delta$.*

It is straightforward to obtain from Theorem 2 the following sample complexity bound.

**Corollary 3.** *With probability $1 - \delta$ the average per-step regret is at most $\varepsilon$ for any*

$$T \ \geq \ c_2 \frac{D^2 S^2 A}{\varepsilon^2} \log \left( \frac{DSA}{\delta \varepsilon} \right)$$

*steps, where $c_2$ is a constant independent of $M$.*

**Remark 4.** *The proof method of Theorem 2 can be modified to give for each initial state $s$ and $T > 1$ an alternative upper bound on the expected regret,*

$$\mathbb{E}\left[\Delta(M, \text{UCRL2}, s, T)\right] \leq c_3 \frac{D^2 S^2 A \log T}{g},$$

*where $g := \rho^*(M) - \max_{\pi,s}\{\rho(M, \pi, s) : \rho(M, \pi, s) < \rho^*(M)\}$ is the gap between the optimal average reward and the second best average reward achievable in $M$.*

These new bounds are improvements over the bounds that have been achieved in [5] for the original UCRL algorithm in various respects: the exponents of the relevant parameters have been decreased considerably, the parameter $D$ we use here is substantially smaller than the corresponding mixing time in [5], and finally, the ergodicity assumption is replaced by the much weaker and more natural assumption that the MDP has finite diameter.

The following is an accompanying lower bound on the expected regret.

**Theorem 5.** *For some $c_4 > 0$, any algorithm $\mathfrak{A}$, and any natural numbers $S, A \geq 10$, $D \geq 20 \log_A S$, and $T \geq DSA$, there is an MDP [3] $M$ with $S$ states, $A$ actions, and diameter $D$, such that for any initial state $s \in \mathcal{S}$ the expected regret of $\mathfrak{A}$ after $T$ steps is*

$$\mathbb{E}\left[\Delta(M, \mathfrak{A}, s, T)\right] \geq c_4 \cdot \sqrt{DSAT} \ .$$

In a different setting, a modification of UCRL2 can also deal with changing MDPs.

**Remark 6.** *Assume that the MDP (i.e. its transition probabilities and reward distributions) is allowed to change $\ell$ times up to step $T$, such that the diameter is always at most $D$ (we assume an initial change at time $t = 1$). In this model we measure regret as the sum of missed rewards compared to the $\ell$ policies which are optimal after the changes of the MDP. Restarting UCRL2 with parameter $\delta/\ell^2$ at steps $\lceil i^3/\ell^2 \rceil$ for $i = 1, 2, 3 \ldots$, this regret is upper bounded by*

$$c_5 \cdot \ell^{\frac{1}{3}} \, T^{\frac{2}{3}} \, DS \, \sqrt{A \log \tfrac{T}{\delta}}$$

*with probability $1 - 2\delta$.*

MDPs with a different model of changing rewards have already been considered in [12]. There, the transition probabilities are assumed to be fixed and known to the learner, but the rewards are allowed to change in every step. A best possible upper bound of $O(\sqrt{T})$ on the regret against an optimal stationary policy, given all the reward changes in advance, is derived.

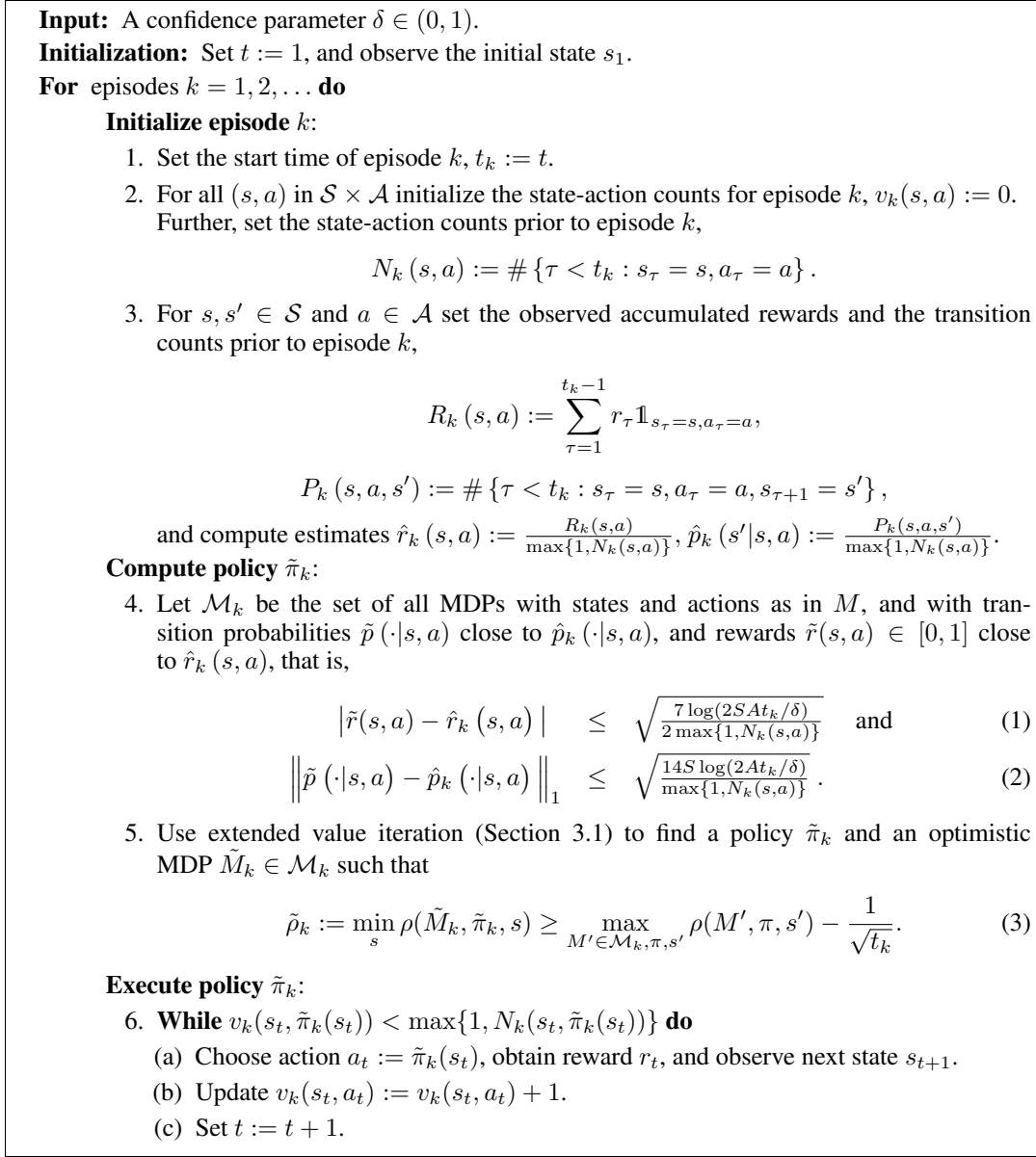

**Input:** A confidence parameter $\delta \in (0, 1)$.

**Initialization:** Set $t := 1$, and observe the initial state $s_1$.

**For** episodes $k = 1, 2, \ldots$ **do**

**Initialize episode** $k$:

1. Set the start time of episode $k$, $t_k := t$.

2. For all $(s, a)$ in $\mathcal{S} \times \mathcal{A}$ initialize the state-action counts for episode $k$, $v_k(s, a) := 0$. Further, set the state-action counts prior to episode $k$,
$$N_k(s, a) := \# \{\tau < t_k : s_\tau = s, a_\tau = a\}.$$

3. For $s, s' \in \mathcal{S}$ and $a \in \mathcal{A}$ set the observed accumulated rewards and the transition counts prior to episode $k$,
$$R_k(s, a) := \sum_{\tau=1}^{t_k-1} r_\tau \mathbb{1}_{s_\tau = s, a_\tau = a},$$
$$P_k(s, a, s') := \# \{\tau < t_k : s_\tau = s, a_\tau = a, s_{\tau+1} = s'\},$$
and compute estimates $\hat{r}_k(s, a) := \frac{R_k(s,a)}{\max\{1, N_k(s,a)\}}$, $\hat{p}_k(s'|s, a) := \frac{P_k(s,a,s')}{\max\{1, N_k(s,a)\}}$.

**Compute policy** $\tilde{\pi}_k$:

4. Let $\mathcal{M}_k$ be the set of all MDPs with states and actions as in $M$, and with transition probabilities $\tilde{p}(\cdot|s, a)$ close to $\hat{p}_k(\cdot|s, a)$, and rewards $\tilde{r}(s, a) \in [0, 1]$ close to $\hat{r}_k(s, a)$, that is,
$$\left| \tilde{r}(s, a) - \hat{r}_k(s, a) \right| \leq \sqrt{\frac{7 \log(2SAt_k/\delta)}{2 \max\{1, N_k(s,a)\}}} \quad \text{and} \tag{1}$$
$$\left\| \tilde{p}(\cdot|s, a) - \hat{p}_k(\cdot|s, a) \right\|_1 \leq \sqrt{\frac{14S \log(2At_k/\delta)}{\max\{1, N_k(s,a)\}}}. \tag{2}$$

5. Use extended value iteration (Section 3.1) to find a policy $\tilde{\pi}_k$ and an optimistic MDP $\tilde{M}_k \in \mathcal{M}_k$ such that
$$\tilde{\rho}_k := \min_s \rho(\tilde{M}_k, \tilde{\pi}_k, s) \geq \max_{M' \in \mathcal{M}_k, \pi, s'} \rho(M', \pi, s') - \frac{1}{\sqrt{t_k}}. \tag{3}$$

**Execute policy** $\tilde{\pi}_k$:

6. **While** $v_k(s_t, \tilde{\pi}_k(s_t)) < \max\{1, N_k(s_t, \tilde{\pi}_k(s_t))\}$ **do**

   (a) Choose action $a_t := \tilde{\pi}_k(s_t)$, obtain reward $r_t$, and observe next state $s_{t+1}$.

   (b) Update $v_k(s_t, a_t) := v_k(s_t, a_t) + 1$.

   (c) Set $t := t + 1$.

Figure 1: The UCRL2 algorithm.

## 3 The UCRL2 Algorithm

Our algorithm is a variant of the UCRL algorithm in [5]. As its predecessor, UCRL2 implements the paradigm of "optimism in the face of uncertainty". As such, it defines a set $\mathcal{M}$ of statistically *plausible* MDPs given the observations so far, and chooses an optimistic MDP $\tilde{M}$ (with respect to the achievable average reward) among these plausible MDPs. Then it executes a policy $\tilde{\pi}$ which is (nearly) optimal for the optimistic MDP $\tilde{M}$.

More precisely, UCRL2 (Figure 1) proceeds in episodes and computes a new policy $\tilde{\pi}_k$ only at the beginning of each episode $k$. The lengths of the episodes are not fixed a priori, but depend on the observations made. In Steps 2–3, UCRL2 computes estimates $\hat{p}_k(s'|s, a)$ and $\hat{r}_k(s, a)$ for the transition probabilities and mean rewards from the observations made before episode $k$. In Step 4, a set $\mathcal{M}_k$ of plausible MDPs is defined in terms of confidence regions around the estimated mean rewards $\hat{r}_k(s, a)$ and transition probabilities $\hat{p}_k(s'|s, a)$. This guarantees that with high probability

the true MDP $M$ is in $\mathcal{M}_k$. In Step 5, *extended value iteration* (see below) is used to choose a near-optimal policy $\tilde{\pi}_k$ on an optimistic MDP $\tilde{M}_k \in \mathcal{M}_k$. This policy $\tilde{\pi}_k$ is executed throughout episode $k$ (Step 6). Episode $k$ ends when a state $s$ is visited in which the action $a = \tilde{\pi}_k(s)$ induced by the current policy has been chosen *in* episode $k$ equally often as *before* episode $k$. Thus, the total number of occurrences of any state-action pair is at most doubled during an episode. The counts $v_k(s, a)$ keep track of these occurrences in episode $k$.[4]

### 3.1 Extended Value Iteration

In Step 5 of the UCRL2 algorithm we need to find a near-optimal policy $\tilde{\pi}_k$ for an optimistic MDP. While value iteration typically calculates a policy for a fixed MDP, we also need to select an optimistic MDP $\tilde{M}_k$ which gives almost maximal reward among all plausible MDPs. This can be achieved by extending value iteration to search also among the plausible MDPs. Formally, this can be seen as undiscounted value iteration [4] on an MDP with extended action set. We denote the state values of the $i$-th iteration by $u_i(s)$ and the normalized state values by $u_i'(s)$ and get for all $s \in \mathcal{S}$:

$$u_0(s) = 0,$$

$$u_{i+1}(s) = \max_{a \in \mathcal{A}} \left\{ \tilde{r}_k(s, a) + \max_{p(\cdot) \in \mathcal{P}(s,a)} \left\{ \sum_{s' \in \mathcal{S}} p(s') \cdot u_i(s') \right\} \right\}, \tag{4}$$

Here $\tilde{r}_k(s, a)$ are the maximal rewards satisfying condition (1) in algorithm UCRL2, and $\mathcal{P}(s, a)$ is the set of transition probabilities $\tilde{p}(\cdot|s, a)$ satisfying condition (2).

While (4) may look like a step of value iteration with an infinite action space, $\max_p \boldsymbol{p} \cdot \boldsymbol{u}_i$ is actually a linear optimization problem over the convex polytope $\mathcal{P}(s, a)$. This implies that only the finite number of vertices of the polytope need to be considered as extended actions, which guarantees convergence of the value iteration.[5]

The value iteration is stopped when

$$\max_{s \in S} \left\{ u_{i+1}(s) - u_i(s) \right\} - \min_{s \in S} \left\{ u_{i+1}(s) - u_i(s) \right\} < \frac{1}{\sqrt{t_k}}, \tag{5}$$

which means that the change of the state values is almost uniform and actually close to the average reward of the optimal policy. It can be shown that the actions, rewards, and transition probabilities chosen in (4) for this $i$-th iteration define an optimistic MDP $\tilde{M}_k$ and a policy $\tilde{\pi}_k$ which satisfy condition (3) of algorithm UCRL2.

## 4 Analysis of UCRL2 and Proof Sketch of Theorem 2

In the following we present an outline of the main steps of the proof of Theorem 2. Details and the complete proofs can be found in the full version of the paper [13]. We also make the assumption that the rewards $r(s, a)$ are deterministic and known to the learner.[6] This simplifies the exposition. Considering unknown stochastic rewards adds little to the proof and only lower order terms to the regret bounds. We also assume that the true MDP $M$ satisfies the confidence bounds in Step 4 of algorithm UCRL2 such that $M \in \mathcal{M}_k$. This can be shown to hold with sufficiently high probability (using a union bound over all $T$).

We start by considering the regret in a single episode $k$. Since the optimistic average reward $\tilde{\rho}_k$ of the optimistically chosen policy $\tilde{\pi}_k$ is essentially larger than the true optimal average reward $\rho^*$, it is sufficient to calculate by how much the optimistic average reward $\tilde{\rho}_k$ overestimates the actual rewards of policy $\tilde{\pi}_k$. By the choice of $\tilde{\pi}_k$ and $\tilde{M}_k$ in Step 5 of UCRL2, $\tilde{\rho}_k \geq \rho^* - 1/\sqrt{t_k}$. Thus the

regret $\Delta_k$ during episode $k$ is bounded as

$$\Delta_k := \sum_{t=t_k}^{t_{k+1}-1} (\rho^* - r_t) \leq \sum_{t=t_k}^{t_{k+1}-1} (\tilde{\rho}_k - r_t) + \frac{t_{k+1} - t_k}{\sqrt{t_k}} \ .$$

The sum over $k$ of the second term on the right hand side is $O(\sqrt{T})$ and will not be considered further in this proof sketch. The first term on the right hand side can be rewritten using the known deterministic rewards $r(s, a)$ and the occurrences of state action pairs $(s, a)$ in episode $k$,

$$\Delta_k \lesssim \sum_{t=t_k}^{t_{k+1}-1} (\tilde{\rho}_k - r_t) = \sum_{(s,a)} v_k(s, a)(\tilde{\rho}_k - r(s, a)). \tag{6}$$

## 4.1 Extended Value Iteration revisited

To proceed, we reconsider the extended value iteration in Section 3.1. As an important observation for our analysis, we find that for any iteration $i$ the range of the state values is bounded by the diameter of the MDP $M$,

$$\max_s u_i(s) - \min_s u_i(s) \leq D. \tag{7}$$

To see this, observe that $u_i(s)$ is the total expected reward after $i$ steps of an optimal non-stationary $i$-step policy starting in state $s$, on the MDP with extended action set as considered for the extended value iteration. The diameter of this extended MDP is at most $D$ as it contains the actions of the true MDP $M$. If there were states with $u_i(s_1) - u_i(s_0) > D$, then an improved value for $u_i(s_0)$ could be achieved by the following policy: First follow a policy which moves from $s_0$ to $s_1$ most quickly, which takes at most $D$ steps on average. Then follow the optimal $i$-step policy for $s_1$. Since only $D$ of the $i$ rewards of the policy for $s_1$ are missed, this policy gives $u_i(s_0) \geq u_i(s_1) - D$, proving (7).

For the convergence criterion (5) it can be shown that at the corresponding iteration

$$|u_{i+1}(s) - u_i(s) - \tilde{\rho}_k| \leq \frac{1}{\sqrt{t_k}}$$

for all $s \in \mathcal{S}$, where $\tilde{\rho}_k$ is the average reward of the policy $\tilde{\pi}_k$ chosen in this iteration on the optimistic MDP $\tilde{M}_k$.[7] Expanding $u_{i+1}(s)$ according to (4), we get

$$u_{i+1}(s) = r(s, \tilde{\pi}_k(s)) + \sum_{s'} \tilde{p}_k (s'|s, \tilde{\pi}_k(s)) \cdot u_i(s')$$

and hence

$$\left| \left( \tilde{\rho}_k - r(s, \tilde{\pi}_k(s)) \right) - \left( \sum_{s'} \tilde{p}_k (s'|s, \tilde{\pi}_k(s)) \cdot u_i(s') - u_i(s) \right) \right| \leq \frac{1}{\sqrt{t_k}}.$$

Defining $\boldsymbol{r}_k := \left( r_k\big(s, \tilde{\pi}_k(s)\big) \right)_s$ as the (column) vector of rewards for policy $\tilde{\pi}_k$, $\tilde{\boldsymbol{P}}_k := \left( \tilde{p}_k (s'|s, \tilde{\pi}_k(s)) \right)_{s,s'}$ as the transition matrix of $\tilde{\pi}_k$ on $\tilde{M}_k$, and $\boldsymbol{v}_k := \left( v_k\big(s, \tilde{\pi}_k(s)\big) \right)_s$ as the (row) vector of visit counts for each state and the corresponding action chosen by $\tilde{\pi}_k$, we can rewrite (6) as

$$\Delta_k \lesssim \sum_{(s,a)} v_k(s, a)(\tilde{\rho}_k - r(s, a)) \leq \boldsymbol{v}_k\big(\tilde{\boldsymbol{P}}_k - \boldsymbol{I}\big)\boldsymbol{u}_i + \sum_{(s,a)} \frac{v_k(s, a)}{\sqrt{t_k}}, \tag{8}$$

recalling that $v_k(s, a) = 0$ for $a \neq \tilde{\pi}_k(s)$. Since the rows of $\tilde{\boldsymbol{P}}_k$ sum to 1, we can replace $\boldsymbol{u}_i$ by $\boldsymbol{w}_k$ with $w_k(s) = u_i(s) - \min_s u_i(s)$ (we again use the subscript $k$ to reference the episode). The last term on the right hand side of (8) is of lower order, and by (7) we have

$$\Delta_k \lesssim \boldsymbol{v}_k\big(\tilde{\boldsymbol{P}}_k - \boldsymbol{I}\big)\boldsymbol{w}_k, \tag{9}$$

$$\|\boldsymbol{w}_k\|_\infty \leq D. \tag{10}$$

## 4.2 Completing the Proof

Replacing the transition matrix $\tilde{\boldsymbol{P}}_k$ of the policy $\tilde{\pi}_k$ in the optimistic MDP $\tilde{M}_k$ by the transition matrix $\boldsymbol{P}_k$ of $\tilde{\pi}_k$ in the true MDP $M$, we get

$$
\begin{aligned}
\Delta_k \;\lesssim\;& \boldsymbol{v}_k\big(\tilde{\boldsymbol{P}}_k - \boldsymbol{I}\big)\boldsymbol{w}_k = \boldsymbol{v}_k\big(\tilde{\boldsymbol{P}}_k - \boldsymbol{P}_k + \boldsymbol{P}_k - \boldsymbol{I}\big)\boldsymbol{w}_k \\
=\;& \boldsymbol{v}_k\big(\tilde{\boldsymbol{P}}_k - \boldsymbol{P}_k\big)\boldsymbol{w}_k + \boldsymbol{v}_k\big(\boldsymbol{P}_k - \boldsymbol{I}\big)\boldsymbol{w}_k.
\end{aligned} \tag{11}
$$

The intuition about the second term in (11) is that the counts of the state visits $\boldsymbol{v}_k$ are relatively close to the stationary distribution of the transition matrix $\boldsymbol{P}_k$, such that $\boldsymbol{v}_k\big(\boldsymbol{P}_k - \boldsymbol{I}\big)$ should be small. The formal proof requires the definition of a suitable martingale and the use of concentration inequalities for this martingale. This yields

$$
\sum_k \boldsymbol{v}_k\big(\boldsymbol{P}_k - \boldsymbol{I}\big)\boldsymbol{w}_k = O\left(D\sqrt{T\log\frac{T}{\delta}}\right)
$$

with high probability, which gives a lower order term in our regret bound. Thus, our regret bound is mainly determined by the first term in (11). Since $\tilde{M}_k$ and $M$ are in the set of plausible MDPs $\mathcal{M}_k$, this term can be bounded using condition (2) in algorithm UCRL2:

$$
\begin{aligned}
\Delta_k \lesssim \boldsymbol{v}_k\big(\tilde{\boldsymbol{P}}_k - \boldsymbol{P}_k\big)\boldsymbol{w}_k \;=\;& \sum_s \sum_{s'} v_k\big(s, \tilde{\pi}_k(s)\big) \cdot \big(\tilde{\boldsymbol{P}}_k(s, s') - \boldsymbol{P}_k(s, s')\big) \cdot w_k(s') \\
\leq\;& \sum_s v_k\big(s, \tilde{\pi}_k(s)\big) \cdot \left\|\tilde{\boldsymbol{P}}_k(s, \cdot) - \boldsymbol{P}_k(s, \cdot)\right\|_1 \cdot \|\boldsymbol{w}_k\|_\infty \\
\leq\;& \sum_s v_k\big(s, \tilde{\pi}_k(s)\big) \cdot 2\sqrt{\frac{14S\log(2AT/\delta)}{\max\{1, N_k(s, \tilde{\pi}_k(s))\}}} \cdot D \;.
\end{aligned} \tag{12}
$$

Let $N(s, a) := \sum_k v_k(s, a)$ such that $\sum_{(s,a)} N(s, a) = T$ and recall that $N_k(s, a) = \sum_{i<k} v_i(s, a)$. By the condition of the while-loop in Step 6 of algorithm UCRL2, we have that $v_k(s, a) \leq N_k(s, a)$. Summing (12) over all episodes $k$ we get

$$
\begin{aligned}
\sum_k \Delta_k \;\leq\;& const \cdot \sum_k \sum_{(s,a)} v_k(s, a) \cdot \sqrt{\frac{S\log(AT/\delta)}{\max\{1, N_k(s,a)\}}} \cdot D \\
=\;& const \cdot D \cdot \sqrt{S\log(AT/\delta)} \cdot \sum_{(s,a)} \sum_k \frac{v_k(s,a)}{\sqrt{\max\{1, N_k(s,a)\}}} \\
\leq\;& const \cdot D \cdot \sqrt{S\log(AT/\delta)} \cdot \sum_{(s,a)} \sqrt{N(s, a)} \tag{13} \\
\leq\;& const \cdot D \cdot \sqrt{S\log(AT/\delta)} \cdot \sqrt{SAT}. \tag{14}
\end{aligned}
$$

Here we used for (13) that

$$
\sum_{k=1}^n \frac{x_k}{\sqrt{X_{k-1}}} \leq \left(\sqrt{2} + 1\right)\sqrt{X_n}\,,
$$

where $X_k = \max\left\{1, \sum_{i=1}^k x_i\right\}$ and $0 \leq x_k \leq X_{k-1}$, and we used Jensen's inequality for (14). Noting that Theorem 2 holds trivially true for $T \leq A$ gives the bound of the theorem.

## 5  The Lower Bound (Proof Sketch for Theorem 5)

We first consider an MDP with two states $s_0$ and $s_1$, and $A' = \lfloor (A - 1)/2 \rfloor$ actions. For each action $a$, let $r(s_0, a) = 0$, $r(s_1, a) = 1$, and $p(s_0|s_1, a) = \delta$ where $\delta = 10/D$. For all but a single "good" action $a^*$ let $p(s_1|s_0, a) = \delta$, while $p(s_1|s_0, a^*) = \delta + \varepsilon$ for some $0 < \varepsilon < \delta$. The diameter of this MDP is $1/\delta$. The average reward of a policy which chooses action $a^*$ in state $s_0$ is $\frac{\delta+\varepsilon}{2\delta+\varepsilon} > \frac{1}{2}$, while the average reward of any other policy is $\frac{1}{2}$. Thus the regret suffered by a suboptimal action in state $s_0$ is $\Omega(\varepsilon/\delta)$. The main observation for the proof of the lower bound is that any algorithm

needs to probe $\Omega(A')$ actions in state $s_0$ for $\Omega(\delta/\varepsilon^2)$ times on average, to detect the "good" action $a^*$ reliably.

Considering $k := \lfloor S/2 \rfloor$ copies of this MDP where only one of the copies has such a "good" action $a^*$, we find that $\Omega(kA')$ actions in the $s_0$-states of the copies need to be probed for $\Omega(\delta/\varepsilon^2)$ times to detect the "good" action. Setting $\varepsilon = \sqrt{\delta k A'/T}$, suboptimal actions need to be taken $\Omega(kA'\delta/\varepsilon^2) = \Omega(T)$ times which gives $\Omega(T\varepsilon/\delta) = \Omega(\sqrt{TDSA})$ regret.

Finally, we need to connect the $k$ copies into a single MDP. This can be done by introducing $A'+1$ additional deterministic actions per state, which do not leave the $s_1$-states but connect the $s_0$-states of the $k$ copies by inducing an $A'$-ary tree structure on the $s_0$-states (1 action for going toward the root, $A'$ actions to go toward the leaves). The diameter of the resulting MDP is at most $2(D/10 + \lceil \log_{A'} k \rceil)$ which is twice the time to travel to or from the root for any state in the MDP. Thus we have constructed an MDP with $\leq S$ states, $\leq A$ actions, and diameter $\leq D$ which forces regret $\Omega(\sqrt{DSAT})$ on any algorithm. This proves the theorem.

### Acknowledgments

This work was supported in part by the Austrian Science Fund FWF (S9104-N13 SP4). The research leading to these results has received funding from the European Community's Seventh Framework Programme (FP7/2007-2013) under grant agreements n° 216886 (PASCAL2 Network of Excellence), and n° 216529 (Personal Information Navigator Adapting Through Viewing, PinView). This publication only reflects the authors' views.

## Footnotes

[1]It can be shown that $\max_{\mathfrak{A}} \mathbb{E}\left[R(M, \mathfrak{A}, s, T)\right] = T\rho^*(M) + O(D(M))$ and $\max_{\mathfrak{A}} R(M, \mathfrak{A}, s, T) = T\rho^*(M) + \tilde{O}(\sqrt{T})$ with high probability.

[2]Actually, the state values.

[3] The diameter of any MDP with $S$ states and $A$ actions is at least $\log_A S$.

[4]Since the policy $\tilde{\pi}_k$ is fixed for episode $k$, $v_k(s, a) \neq 0$ only for $a = \tilde{\pi}_k(s)$. Nevertheless, we find it convenient to use a notation which explicitly includes the action $a$ in $v_k(s, a)$.

[5]Because of the special structure of the polytope $\mathcal{P}(s, a)$, the linear program in (4) can be solved very efficiently in $O(S)$ steps after sorting the state values $u_i(s')$. For the formal convergence proof also the periodicity of optimal policies in the extended MDP needs to be considered.

[6]In this case all plausible MDPs considered in Steps 4 and 5 of algorithm UCRL2 would give these rewards.

[7]This is quite intuitive. We expect to receive average reward $\tilde{\rho}_k$ per step, such that the difference of the state values after $i + 1$ and $i$ steps should be about $\tilde{\rho}_k$.

### References

[1] Richard S. Sutton and Andrew G. Barto. *Reinforcement Learning: An Introduction*. MIT Press, 1998.

[2] Michael J. Kearns and Satinder P. Singh. Finite-sample convergence rates for Q-learning and indirect algorithms. In *Advances in Neural Information Processing Systems 11*. MIT Press, 1999.

[3] Michael J. Kearns and Satinder P. Singh. Near-optimal reinforcement learning in polynomial time. *Mach. Learn.*, 49:209–232, 2002.

[4] Martin L. Puterman. *Markov Decision Processes: Discrete Stochastic Dynamic Programming*. John Wiley & Sons, Inc., New York, NY, USA, 1994.

[5] Peter Auer and Ronald Ortner. Logarithmic online regret bounds for reinforcement learning. In *Advances in Neural Information Processing Systems 19*, pages 49–56. MIT Press, 2007.

[6] Ronen I. Brafman and Moshe Tennenholtz. R-max – a general polynomial time algorithm for near-optimal reinforcement learning. *J. Mach. Learn. Res.*, 3:213–231, 2002.

[7] Ambuj Tewari and Peter Bartlett. Optimistic linear programming gives logarithmic regret for irreducible mdps. In *Advances in Neural Information Processing Systems 20*, pages 1505–1512. MIT Press, 2008.

[8] Sham M. Kakade. *On the Sample Complexity of Reinforcement Learning*. PhD thesis, University College London, 2003.

[9] Alexander L. Strehl and Michael L. Littman. A theoretical analysis of model-based interval estimation. In *Proc. 22nd ICML 2005*, pages 857–864, 2005.

[10] Alexander L. Strehl and Michael L. Littman. An analysis of model-based interval estimation for Markov decision processes. *J. Comput. System Sci.*, 74(8):1309–1331, 2008.

[11] Apostolos N. Burnetas and Michael N. Katehakis. Optimal adaptive policies for Markov decision processes. *Math. Oper. Res.*, 22(1):222–255, 1997.

[12] Eyal Even-Dar, Sham M. Kakade, and Yishay Mansour. Experts in a Markov decision process. In *Advances in Neural Information Processing Systems 17*, pages 401–408. MIT Press, 2005.

[13] Peter Auer, Thomas Jaksch, and Ronald Ortner. Near-optimal regret bounds for reinforcement learning. Technical Report CIT-2009-01, University of Leoben, Chair for Information Technology, 2009. http://institute.unileoben.ac.at/infotech/publications/TR/CIT-2009-01.pdf.
